# A Self-Organizing Integrated Segmentation And Recognition Neural Net

**Jim Keeler** *
MCC
3500 West Balcones Center Drive
Austin, TX 78729

**David E. Rumelhart**
Psychology Department
Stanford University
Stanford, CA 94305

## Abstract

We present a neural network algorithm that simultaneously performs segmentation and recognition of input patterns that self-organizes to detect input pattern locations and pattern boundaries. We demonstrate this neural network architecture on character recognition using the NIST database and report on results herein. The resulting system simultaneously segments and recognizes touching or overlapping characters, broken characters, and noisy images with high accuracy.

## 1 INTRODUCTION

Standard pattern recognition systems usually involve a segmentation step prior to the recognition step. For example, it is very common in character recognition to segment characters in a pre-processing step then normalize the individual characters and pass them to a recognition engine such as a neural network, as in the work of LeCun et al. 1988, Martin and Pittman (1988).

This separation between segmentation and recognition becomes unreliable if the characters are touching each other, touching bounding boxes, broken, or noisy. Other applications such as scene analysis or continuous speech recognition pose similar and more severe segmentation problems. The difficulties encountered in these applications present an apparent dilemma: one cannot recognize the patterns

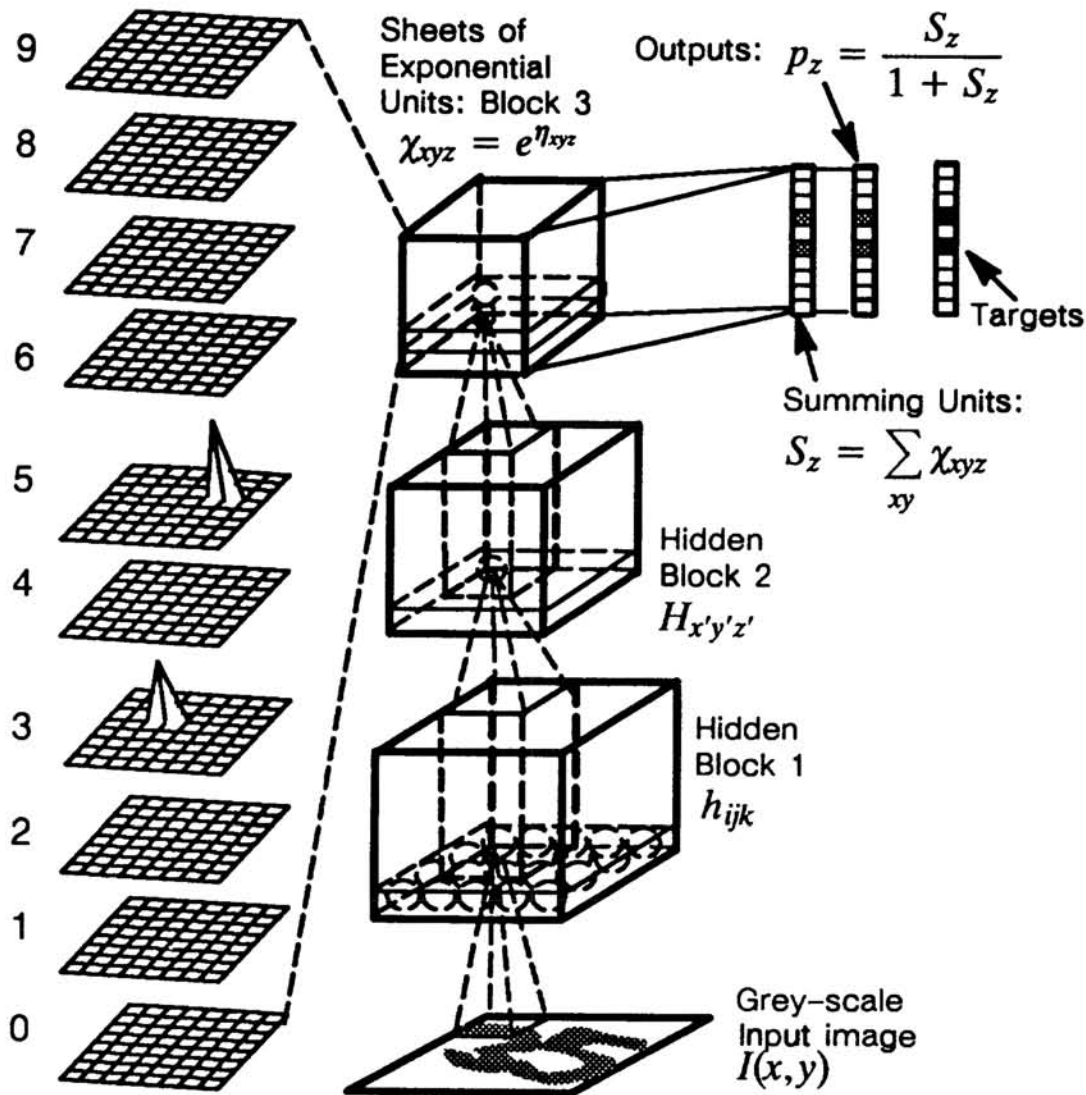

Figure 1: The ISR network architecture. The input image may contain several characters and is presented to the network in a two-dimensional grey-scale image. The units in the first block, $h_{ijk}$, have linked-local receptive field connections to the input image. Block 2, $H_{x'y'z'}$, has a three-dimensional linked-local receptive field to block 1, and the exponential unit block, block 3, has three-dimensional linked-local receptive field connections to block 2. These linked fields insure translational invariance (except for edge-effects at the boundary). The exponential unit block has one layer for each output category. These units are the output units in the test mode, but hidden units during training: the exponential unit activity is summed over $(s_z)$ to project out the positional information, then converted to a probability $p_z$. Once trained, the exponential unit layers serve as "smart histograms" giving sharp peaks of activity directly above the corresponding characters in the input image, as shown to the left.

until they are segmented, yet in many cases one cannot segment the patterns until they are recognized.

A solution to this apparent dilemm is to simultaneously segment and recognize the patterns. Integration of the segmentation and recognition steps is essential for further progress in these difficult pattern recognition tasks, and much effort has been devoted to this topic in speech recognition. For example, Hidden Markov models integrate the task of segmentation and recognition as a part of the word-recognition module. Nevertheless, little neural network research in pattern recognition has focused on the integrated segmentation and recognition (ISR) problem.

There are several ways to achieve ISR in a neural network. The first use of back-propagation ISR neural networks for character recognition was reported by Keeler, Rumelhart and Leow (1991a). The ISR neural network architecture is similar to the time-delayed neural network architecture for speech recognition used by Lang, Hinton, and Waibel (1990).

The following section outlines the neural network algorithm and architecture. Details and rationale for the exact structure and assumptions of the network can be found in Keeler et al. (1991a,b).

## 2    NETWORK ARCHITECTURE AND ALGORITHM

The basic organization of the network is illustrated in Figure 2. The input consists of a two-dimensional grey-scale image representing the pattern to be processed. We designate this input pattern by the two-dimensional field $I(x, y)$. In general, we assume that any pattern can be presented at any location and that the characters may touch, overlap or be broken or noisy. The input then projects to a linked-local-receptive-field block of sigmoidal hidden units (to enforce translational invariance). We designate the activation of the sigmoidal units in this block by $h_{ijk}$.

The second block of hidden units, $H_{x'y'z'}$, is a linked-local receptive field block of sigmoidal units that receives input from a three-dimensional receptive field in the $h_{ijk}$ block. In a standard neural network architecture we would normally connect block $H$ to the output units. However we connect block $H$ to a block of exponential units $\chi_{xyz}$. The $\chi$ block serves as the outputs after the network has been trained; there is a sheet of exponential units for each output category. These units are connected to block H via a linked-local receptive field structure. $\chi_{xyz} = e^{\eta_{xyz}}$, where the net input to the unit is

$$\eta_{xyz} = \sum_{x'y'} W^{xyz}_{x'y'z'} H_{x'y'z'} + \beta_z, \tag{1}$$

and $W^{xyz}_{x'y'z'}$ is the weight from hidden unit $H_{x'y'z'}$ to the exponential unit $\chi_{xyz}$. Since we use linked weights in each block, the entire structure is translationally invariant. We make use of this property in our training algorithm and project out the positional information by summing over the entire layer, $S_z = \sum_{xy} \chi_{xyz}$. This allows us to give non-specific target information in the form of "the input contains a 5 and a 3, but I will not say where." We do this by converting the summed information into an output probability, $p_z = \frac{S_z}{1+S_z}$.

## 2.1   The learning Rule

There are two objective functions that we have used to train ISR networks: cross entropy and total-sum-square-error. $l = \sum_z t_z ln p_z + (1 - t_z) ln(1 - p_z)$, where $t_z$ equals 1 if pattern $z$ is presented and 0 otherwise. Computing the gradient with respect to the net input to a particular exponential unit yields the following term in our learning rule:

$$\frac{\partial l}{\partial \eta_{xyz}} = (t_z - p_z)\frac{\chi_{xyz}}{\sum_{xy} \chi_{xyz}}. \qquad (2)$$

It should be noted that this is a kind of *competitive* rule in which the learning is proportional to the relative strength of the activation at the unit at a particular location in the $\chi$ layer to the strength of activation in the entire layer. For example, suppose that $\chi_{2,3,5} = 1000$ and $\chi_{5,3,5} = 100$. Given the above rules, $\chi_{2,3,5}$ would receive about 10 times more of the output error than the unit $\chi_{5,3,5}$. Thus the units compete with each other for the credit or blame of the output, and the "rich get richer" until the proper target is achieved. This favors self-organization of highly localized spikes of activity in the exponential layers directly above the particular character that the exponential layer detects ("smart histograms" as shown in Figure 1). Note that we never give positional information in the network but that the network self-organizes the exponential unit activity to discern the positional information. The second function is the total-sum-square error, $E = \sum_z (t_z - p_z)^2$. For the total-sum-square error measure, the gradient term becomes

$$\frac{\partial E}{\partial \eta_{xyz}} = (t_z - p_z)\frac{\chi_{xyz}}{(1 + \sum_{xy} \chi_{xyz})^2}. \qquad (3)$$

Again this has a competitive term, but the competition is only important for $\chi_{xyz}$ large, otherwise the denominator is dominated by 1 for small $\sum_{xy} \chi_{xyz}$. We used the quadratic error function for the networks reported in the next section.

# 3   NIST DATABASE RECOGNITION

## 3.1   Data

We tested this neural network algorithm on the problem of segmenting and recognizing handwritten numerals from the NIST database. This database contains approximately 273,000 samples of handwritten numerals collected from the Bureau of Census field staff. There were 50 different forms used in the study, each with 33 fields, 28 of which contain handwritten numerals ranging in length from 2 to 10 digits per field. We only used fields of length 2 to 6 (field numbers 6 to 30). We used two test sets: a small test set, Test Set A of approximately 4,000 digits, 1,000 fields, from forms labeled f1800 to f1840 and a larger test set, Test Set B, containing 20,000 numerals 5,000 fields and 200 forms from f1800 to f1899 and f2000 to f2199.

We used two different training sets: a hand-segmented training set containing approximately 33,000 digits from forms f0000 to f0636 (the Segmented Training Set) and another training set that was never hand-segmented from forms f0000 to f1800 (the Unsegmented Training Set. We pre-processed the fields with a simple box-removal and size-normalization program before they were input to the ISR net.

The hand segmentation was conventional in the sense that boxes were drawn around each of the characters, but we the boxes included any other portions of characters that may be nearby or touching in the natural context. Note that precise labeling of the characters is not essential at all. We have trained systems where only the center information the characters was used and found no degradation in performance. This is due to the fact that the system self-organizes the positional information, so it is only required that we know *whether* a character is in a field, not precisely *where*.

## 3.2  TRAINING

We trained several nets on the NIST database. The best training procedure was as follows: Step 1): train the network to an intermediate level of accuracy (96% or so on single characters, about 12 epochs of training set 1). Note that when we train on single characters, we do not need *isolated* characters – there are often portions of other nearby characters within the input field. Indeed, it helps the ISR performance to use this natural context. There are two reasons for this step: the first is speed – training goes much faster with single characters because we can use a small network. We also found a slight generalization accuracy benefit by including this training step. Step 2): copy the weights of this small network into a larger network and start training on 2 and 3 digit fields from the database without hand segmentation. These are fields numbered 6,7,11,15,19,20,23,24,27, and 28. The reason that we use these fields is that we do not have to hand-segment them – we present the fields to the net with the answer that the person was supposed to write in the field. (There were several cases where the person wrote the wrong numbers or didn't write anything. These cases were NOT screened from the training set.) Taking these fields from forms f0000 to f1800 gives us another 45,000 characters to train on without ever segmenting them.

There were several reasons that we use fields of length 2 and 3 and not fields of 4,5,or 6 for training (even though we used these in testing). First, 3 characters covers the most general case: a character either has no characters on either side, one to the left, one to the right or one on both sides (3 characters total). If we train on 3 characters and duplicate the weights, we have covered the most general case for any number of characters, and it is clearly faster to train on shorter fields. Second, training with more characters confuses the net. As pointed out in our previous work (keeler 1991a), the learning algorithm that we use is only valid for one or no characters of a given type presented in the input field. Thus, the field '39541' is ok to train on, but the field '288' violates one of the assumptions of the training rule. In this case the two 8's would be competing with each other for the answer and the rule favors only one winner. Even though this problem occurs 1/10th of the time for two digit fields, it is not serious enough to prevent the net from learning. (Clearly it would not learn fields of length 10 where all of the target units are turned on and there would be no chance for discrimination.) This problem could be avoided by incorporating order information into training and we have proposed several mechanisms for incorporating order information in training, but do not use them in the present system. Note that this biases the training toward the a-priori distribution of characters in the 2 and 3 digit fields, which is a different distribution from that of the testing set.

The two networks that we used had the following architectures: Net1: Input: 28x24

receptive fields 6x6 shift 2x2. hidden 1: 12x11x12 receptive fields 4x4x12 shift 2x2x12. hidden 2: 5x4x18 receptive fields 3x3x18 shift 1x1x18. exponentials (block 3): 3x2x10 10 summing, 10 outputs.

Net2: Input: 28x26 receptive fields 6x6 shift 2x4. hidden 1: 12x6x12 receptive fields 5x4x12 shift 1x2x12. hidden 2: 8x2x18 receptive fields 5x2x18 shift 1x1x18. exponentials (block 3): 4x1x10 10 summing, 10 outputs.

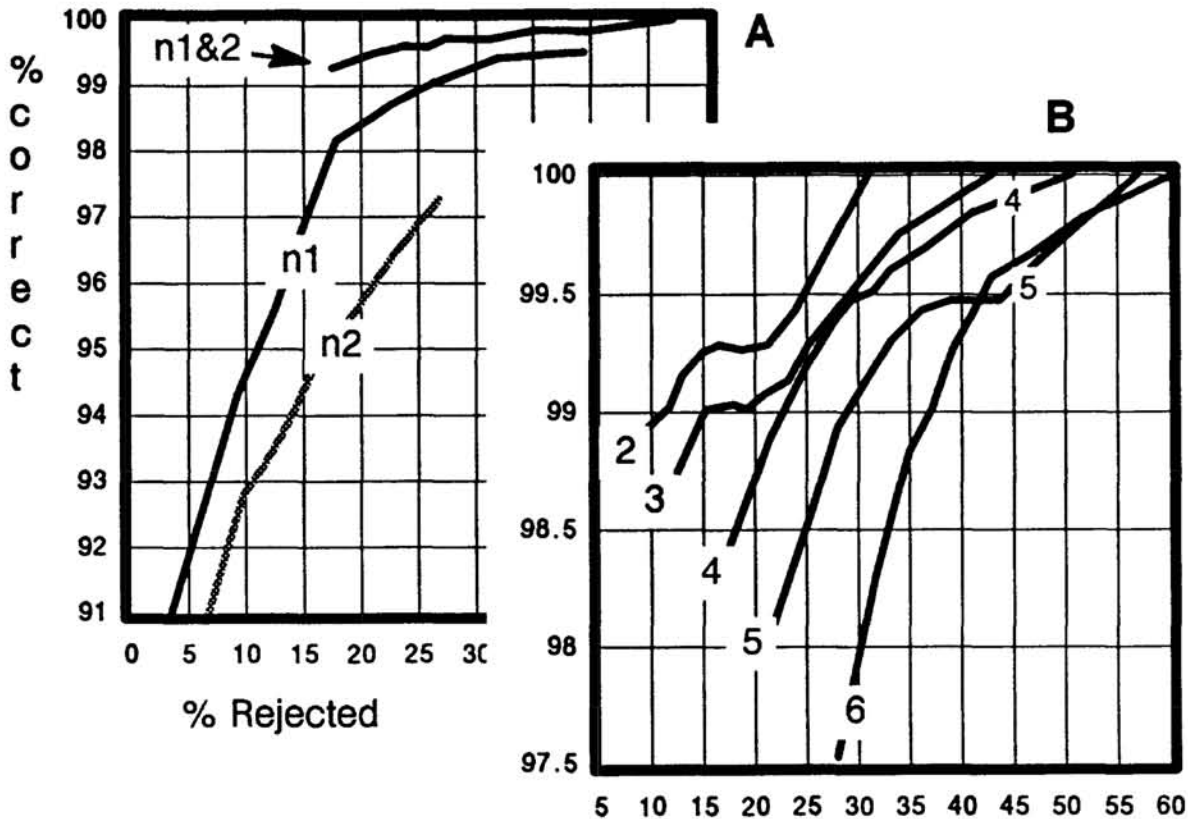

Figure 2: Average combined network performance on the NIST database. Figure 2A shows the generalization performance of two neural networks on the NIST Test Set A. The individual nets Net1 and Net2 (n1, n2 respectively) and the combined performance of nets 1 and 2 are shown where fields are rejected when the nets differ. The curves show results for fields ranging length 2 to 6 averaged over all fields for 1,000 total fields, 4,000 characters. Note that Net2 is not nearly as accurate as Net1 on fields, but that the combination of the two is significantly better than either. For this test set the rejection rate is 17% (83% acceptance) with an accuracy rate of 99.3% (error rate 0.7%) overall on fields of average length 4. Figure 2B shows the per-field performance for test-set B (5,000 fields, 20,000 digits) Again both nets are used for the rejection criterion. For comparison, 99% accuracy on fields of length 4 is achieved at 23% rejection.

Figure 2 shows the generalization performance on the NIST database for Net1, Net2 and their combination. For the combination, we accepted the answer only when the networks agreed and rejected further based on a simple confidence measure (the difference of the two highest activations) of each individual net.

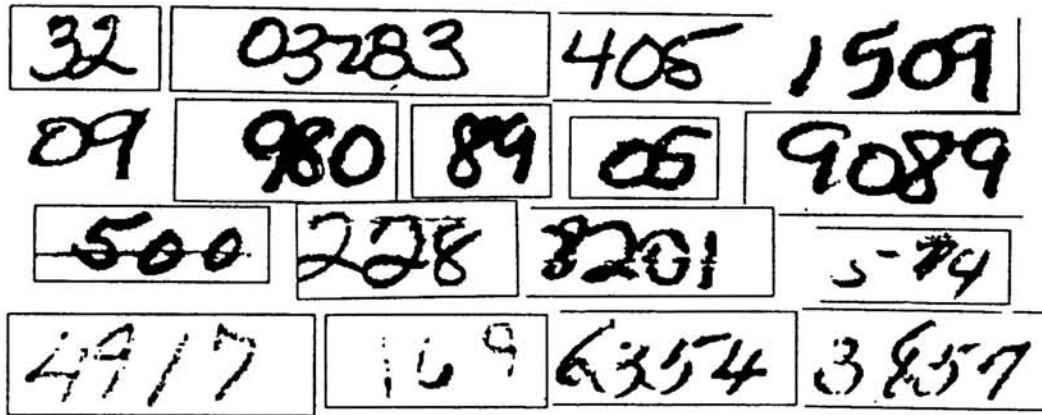

Figure 3: Examples of correctly recognized fields in the NIST database. This figure shows examples of fields that were correctly recognized by the ISR network. Note the cases of touching characters, multiple touching characters, characters touching in multiple places, fields with extrinsic noise, broken characters and touching, broken characters with noise. Because of the integrated nature of the segmentation and recognition, the same system is able to handle all of these cases.

## 4     DISCUSSION AND CONCLUSIONS

This investigation has demonstrated that the ISR algorithm can be used for integrated segmentation and recognition and achieve high-accuracy results on a large database of hand-printed numerals. The overall accuracy rates of 83% acceptance with 99.3% accuracy on fields of average length 4 is competitive with accuracy reported in commercial products. One should be careful making such comparisons. We found a variance of 7% or more in rejection performance on different test sets with more than 1,000 fields (a good statistical sample). Perhaps more important than the high accuracy, we have demonstrated that the ISR system is able to deal with touching, broken and noisy characters. In other investigations we have demonstrated the ISR system on alphabetic characters with good results, and on speech recognition (Keeler, Rumelhart, Zand-Biglari, 1991) where the results are slightly better than Hidden Markov Model results.

There are several attractive aspects about the ISR algorithm: 1) Labeling can be "sloppy" in the sense that the borders of the characters do not have to be defined. This reduces the labor burden of getting a system running. 2) The final weights can be duplicated so that the system can all run in parallel. Even with both networks running, the number of weights and activations needed to be stored in memory is quite small – about 30,000 floating point numbers, and the system is quite fast in the feed-forward mode: peak performance is about 2.5 characters/sec on a Dec 5000 (including everything: both networks running, input pre-processing, parsing the answers, printing results, etc.). This structure is ideal for VLSI implementation since it contains a very small number of weights (about 5,000). This is one possible way around the computational bottleneck facing encountered in processing complex scenes – the ISR net can do very-fast first-cut scene analysis with good discrimi-

nation of similar objects – an extremely difficult task. 3) The ISR algorithm and architecture presents a new and powerful approach of using forward models to convert position-independent training information into position-specific error signals. 4) There is no restriction to one-dimension; The same ISR structure has been used for two-dimensional parsing.

Nevertheless, there are several aspects of the ISR net that require improvement for future progress. First, the algorithmic assumption of having one pattern of a given type in the input field is too restrictive and can cause confusion in some training examples. Second, we are throwing some information away when we project out all of the positional information order information could be incorporated into the training information. This extra information should improve training performance due to the more-specific error signals. Finally, normalization is still a problem. We do a crude normalization, and the networks are able to segment and recognize characters as long as the difference in size is not too large. A factor of two in size difference is easily handled with the ISR system, but a factor of four decreases recognition accuracy by about 3-5% on the character recognition rates. This requires a tighter coupling between the segmentation/recognition and normalization. Just as one must segment and recognize simultaneously, in many cases one can't properly normalize until segmentation/recognition has occurred. Fortunately, in most document processing applications, crude normalization to within a factor of two is simple to achieve, allowing high accuracy networks.

## Acknowledgements

We thank Wee-Kheng Leow, Steve O'Hara, John Canfield, for useful discussions and coding.

## Footnotes

*keeler@mcc.com Reprint requests: coila@mcc.com or at the above address.

## References

[1] J.D. Keeler, D.E. Rumelhart, and W.K. Leow (1991a) "Integrated Segmentation and Recognition of Hand-printed Numerals". In: Lippmann, Moody and Touretzky, Editors, *Neural Information Processing Systems* 3, 557-563.

[2] J.D. Keeler, D.E. Rumelhart, and S. Zand-Biglari (1991b) "A Neural Network For Integrated Segmentation and Recognition of Continuous Speech". MCC Technical Report ACT-NN-359-91.

[3] K. Lang, A. Waibel, G. Hinton. (1990) A time delay Neural Network Architecture for Isolated Word Recognition. *Neural Networks*, 3 23-44.

[4] Y. Le Cun, B. Boser, J.S. Denker, S. Solla, R. Howard, and L. Jackel. (1990) "Back-Propagation applied to Handwritten Zipcode Recognition." *Neural Computation* 1(4):541-551.

[5] G. Martin, J. Pittman (1990) "Recognizing hand-printed letters and digits." In D. Touretzky (Ed.). *Neural Information Processing Systems* 2, 405-414, Morgan Kauffman Publishers, San Mateo, CA.

[6] The NIST database can be obtained by writing to: Standard Reference Data National Institute of Standards and Technology 221/A323 Gaithersburg, MD 20899 USA and asking for NIST special database 1 (HWDB).
